# Recovering Intrinsic Images with a Global Sparsity Prior on Reflectance

**Peter Vincent Gehler**
Max Planck Institut for Informatics
pgehler@mpii.de

**Carsten Rother**
Microsoft Research Cambridge
carrot@microsoft.com

**Martin Kiefel, Lumin Zhang, Bernhard Schölkopf**
Max Planck Institute for Intelligent Systems
{mkiefel,lumin,bs}@tuebingen.mpg.de

## Abstract

We address the challenging task of decoupling material properties from lighting properties given a single image. In the last two decades virtually all works have concentrated on exploiting edge information to address this problem. We take a different route by introducing a new prior on reflectance, that models reflectance values as being drawn from a sparse set of basis colors. This results in a Random Field model with global, latent variables (basis colors) and pixel-accurate output reflectance values. We show that without edge information high-quality results can be achieved, that are on par with methods exploiting this source of information. Finally, we are able to improve on state-of-the-art results by integrating edge information into our model. We believe that our new approach is an excellent starting point for future developments in this field.

## 1 Introduction

The task of recovering intrinsic images is to separate a given input image into its material-dependent properties, known as reflectance or albedo, and its light-dependent properties, such as shading, shadows, specular highlights, and inter-reflection. A successful separation of these properties would be beneficial to a number of computer vision tasks. For example, an image which solely depends on material-dependent properties is helpful for image segmentation and object recognition [11], while a clean image of shading is a valuable input to shape-from-shading algorithms.

As in most previous work in this field, we cast the intrinsic image recovery problem into the following simplified form, where each image pixel is the product of two components:

$$I = sR . \tag{1}$$

Here $I \in \mathbb{R}^3$ is the pixel's color, in RGB space, $R \in \mathbb{R}^3$ is its reflectance and $s \in \mathbb{R}$ its "shading". Note, we use "shading" as a proxy for all light-dependent properties, e.g. shadows. The fact that shading is only a 1D entity imposes some limitations. For example, shading effects stemming from multiple light sources can only be modeled if all light sources have the same color.[1] The goal of this work is to estimate $s$ and $R$ given $I$. This problem is severely under-constraint, with 4 unknowns and 3 constraints for each pixel. Hence, a trivial solution to (1) is, for instance, $I = R, s = 1$ for all pixels. The main focus of this paper is on exploring sensible priors for both shading and reflectance.

Despite the importance of this problem surprisingly little research has been conducted in *recent* years. Most of the inventions were done in the 70s and 80s. The recent comparative study [7] has shown that the simple Retinex method [9] from the 70s is still the top performing approach. Given

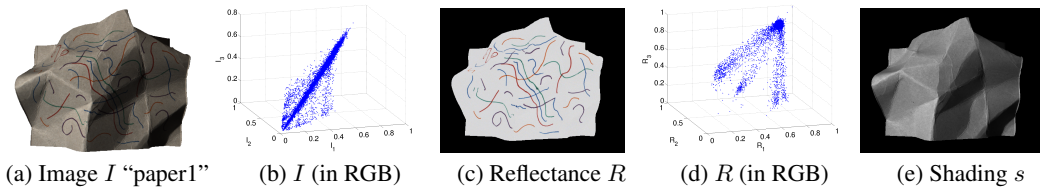

| (a) Image $I$ "paper1" | (b) $I$ (in RGB) | (c) Reflectance $R$ | (d) $R$ (in RGB) | (e) Shading $s$ |

Figure 1: An image (a), its color in RGB space (b), the reflectance image (c), its distribution in RGB space (d), and the shading image (e). Omer and Werman [12] have shown that an image of a natural scene often contains only a few different "basis colorlines". Figure (b) shows a dominant gray-scale color-line and other color lines corresponding to the scribbles on the paper (a). These colorlines are generated by taking a small set of "basis colors" which are then linearly "smeared" out in RGB space. The basis colors are clearly visible in (d), where the cluster for white (top, right) is the dominant one. This "smearing effect" comes from properties of the scene (e.g. shading or shadows), and/or properties of the camera, e.g. motion blur. (Note, the few pixels in-between clusters are due to anti-aliasing effects). In this work we approximate the basis colors by a simple mixture of isotropic Gaussians.

the progress in the last two decades on probabilistic models, inference and learning techniques, as well as the improved computational power, we believe that now is a good time to revisit this problem. This work, together with the recent papers [14, 4, 7, 15], are a first step in this direction.

The main motivation of our work is to develop a simple, yet powerful probabilistic model for shading and reflectance estimation. In total we use three different types of factors. The first one is the most commonly used factor and is key ingredient of all Retinex-based methods. The idea is to extract those image edges which are (potentially) true reflectance edges and then to recover a new reflectance image that contains only these edges, using a set of Poisson equations. This term on its own is enough to recover a non-trivial decomposition, i.e. $s \neq 1$. The next factor is a simple smoothness prior on shading between neighboring image pixels, and has been used by some previous work e.g. [14]. Note, there are a few works, which we discuss in more detail later, that extend these pairwise terms to become patch-based. The third prior term is the main contribution of our work and is conceptually very different from the local (pairwise or patch-based) constraints of previous works. We propose a new global (image-wide) sparsity prior on reflectance based on the findings of [12] and discussed in Fig 1. In the absence of other factors this already produces non-trivial results. This prior takes the form of a Mixture of Gaussians, and encodes the assumption that the reflectance value for each pixel is drawn from some mixing components, which in this context we refer to as "basis colors". The complete model forms a latent variable Random Field model for which we perform MAP estimation.

By combining the different terms we are able to outperform state-of-the art. If we use image optimal parameter settings we perform on par with methods that use multiple images as input. To empirically validate this we use the database introduced in the comparative study [7].

## 2 Related Work

There is a vast amount of literature on the problem of recovering intrinsic images. We refer the reader to detailed surveys in [8, 17, 7], and limit our attention to some few related works.

Barrow and Tenenbaum [2] were the first to define the term "intrinsic image". Around the same time the first solution to this problem was developed by Land and McCann [9] known as the Retinex algorithm. After that the Retinex algorithm was extended to two dimensions by Blake [3] and Horn [8], and later applied to color images [6]. The basic Retinex algorithm is a 2-step procedure: 1) detect all image gradients which are caused by changes in reflectance; 2) recover a reflectance image which preserves the detected reflectance gradients. The basic assumption of this approach is that small image gradients are more likely caused by a shading effect and strong gradients by a change in reflectance. For color images this rule can be extended by treating changes in the 1D brightness domain differently to changes in the 2D chromaticity space.[2] This method, which we denote as "Color Retinex" was the top performing method in the recent comparison paper [7]. Note,

the only approach which could beat Retinex utilizes multiple images [19]. Surprisingly, the study [7] also shows that more sophisticated methods for training the reflectance edge detector, using e.g. images patches, did not perform better than the basic Retinex method. In particular the study tested two methods of Tappen et al. [17, 16]. A plausible explanation is offered, namely that these methods may have over-fitted the small amount of training data. The method [17] has an additional intermediate step where a Markov Random Field (MRF) is used to "propagate" reflectance gradients along contour lines.

The paper [15] implements the same intuition as done here, namely that there is a sparse set of reflectances present in the scene. However both approaches bear the following differences. In [15] a sparsity enforcing term is included, that is penalizing reflectance differences from some prototype references. This term encourages all reflectances to take on the same value, while the model we propose in this paper allows for a mixture of different material reflectances and thus keeps their diversity. Also, in contrast to [15], where a gradient aware wavelet transform is used as a new representation, here we work directly in the RGB domain. By doing so we directly extend previous intrinsic image models which makes evident the gains that can be attributed to a global sparse reflectance term alone.

Recently, Shen et al. [14] introduced an interesting extension of the Retinex method, which bears some similarity with our approach. The key idea in their work is to perform a pre-processing step where the (normalized) reflectance image is partitioned into a few clusters. Each cluster is treated as a non-local "super-pixel". Then a variant of the Retinex method is run on this super-pixel image. The conceptual similarity to our approach is the idea of performing an image-wide clustering step. However, the differences are that they do not formulate this idea as a joint probabilistic model over latent reflectance "basis colors" and shading variables. Furthermore, every pixel in a super-pixel must have the same intensity, which is not the case in our work. Also, they need a Retinex type of edge term to avoid the trivial solution of $s = 1$.

Finally, let us briefly mention techniques which use patch-based constraints, instead of pair-wise terms. The seminal work of Freeman et al. on learning low-level vision [5] formulates a probabilistic model for intrinsic images. In essence, they build a patch-based prior jointly over shading and reflectance. In a new test image the best explanation for reflectance and shading is determined. The key idea is that patches do overlap, and hence form an MRF, where long-range propagation is possible. Since no large-scale ground database was available at that time, they only train and test on computer generated images of blob-like textures. Another patch-based method was recently suggested in [4]. They introduce a new energy term which is satisfied when all reflectance values in a small, e.g. $3 \times 3$, patch lie on a plane in RGB space. This idea is derived from the Laplacian matrix used for image matting [10]. On its own this term gives in practice often the trivial solution $s = 1$. For that reason additional user scribbles are provided to achieve high-quality results.[3]

## 3 A Probabilistic Model for Intrinsic Images

The model outlined here falls into the class of Conditional Random Fields, specifying a conditional probability distribution over reflectance $R$ and shading $S$ components for a given image $I$

$$p(\mathbf{s}, R \mid I) \propto \exp\left(-E(\mathbf{s}, R \mid I)\right). \tag{2}$$

Before we describe the energy function $E$ in detail, let us specify the notation. We will denote with subscripts $i$ the values at location $i$ in the image. Thus $I_i$ is an image pixel (vector of dimension 3), $R_i$ a reflectance vector (a 3-vector), $s_i$ the shading (a scalar). The total number of pixels in an image is $N$. With boldface we denote vectors of components, e.g. $\mathbf{s} = (s_1, \ldots, s_N)$.

There are two ways to use the relationship (1) to formulate a model for shading and reflectance, corresponding to two different image likelihoods $p(I \mid \mathbf{s}, R)$. One possible way is to relax the relation (1) and for example assume a Gaussian likelihood $p(I \mid \mathbf{s}, R) \propto \exp(-\|I - \mathbf{s}R\|^2)$ to account for some noise in the image formation process. This yields an optimization problem with $4N$ unknowns. The second possibility is to assume a delta-prior around $\mathbf{s}R$ which results in the following complexity reduction. Since $I_i^c = s_i R_i^c$ has to hold of all color channels $c = \{R, G, B\}$, the unknown variables are specified up to scalar multipliers, in other words the direction of $R_i$ is already known. We rewrite $R_i = r_i \vec{R}_i$, with $\vec{R}_i = I_i/\|I_i\|$, leaving $\mathbf{r} = (r_1, \ldots, r_N)$ to be the

only unknown variable. The shading components can be computed using $s_i = \|I_i\|/r_i$. Thus the optimization problem is reduced to a search of $N$ variables.

The latter reduction is commonly exploited by intrinsic image algorithms in order to simplify the model [7, 14, 4] and in the remainder we will also make use of it. This allows us to write all model parts in terms of $\mathbf{r}$.

Note that there is a global scalar $k$ by which the result $\mathbf{s}$, $R$ can be modified without effecting eq. (1), i.e. $I = (\mathbf{s}k)(1/kR)$. For visualization purpose $k$ is chosen such that the results are visually closest to the known ground truth.

### 3.1 Model

The energy function we describe here consists of three different terms that are linearly combined. We will describe the three components and their influence in greater detail below, first we write the optimization problem that corresponds to a MAP solution in its most general form

$$\min_{r_i, \alpha_i; i=1,\dots,n} \mathrm{w}_s E_s(\mathbf{r}) + \mathrm{w}_r E_{ret}(\mathbf{r}) + \mathrm{w}_{cl} E_{cl}(\mathbf{r}, \alpha). \tag{3}$$

Note, the global scale of the energy is not important, hence we can always fix one non-zero weight $\mathrm{w}_s, \mathrm{w}_r, \mathrm{w}_{cl}$ to 1.

**Shading Prior ($E_s$)** We expect the shading of an image to vary smoothly over the image and we encode this in the following pairwise factors

$$E_s(r) = \sum_{i \sim j} \left( r_i^{-1}\|I_i\| - r_j^{-1}\|I_j\| \right)^2, \tag{4}$$

where we use a 4-connected pixel graph to encode the neighborhood relation which we denote with $i \sim j$. Because of the dependency on the inverse of $r$, this term is not jointly convex in $\mathbf{r}$. Any model that includes this smoothness prior thus has the (potential) problem of multiple local minima. Empirically we have seen that, however, this function seems to be very well behaved, a large range of different starting points for $\mathbf{r}$ resulted in the same minimum. Nevertheless, we use multiple restarts with different starting points, see optimization selection 3.2.

**Gradient Consistency ($E_{ret}$)** As discussed in the introduction, the main idea of the Retinex algorithm is to disambiguate between edges that are due to shading variations from those that are caused by material reflectance changes. This idea is then implemented as follows. Assume that we already know, or have classified, that an edge at location $i, j$ in the input image is caused by a change in reflectance. Then we know the magnitude of the gradient that has to appear in the reflectance map by noting that $\log(I_i) - \log(I_j) = \log(r_i\vec{R}_i) - \log(r_j\vec{R}_j)$. Using the fact $\log(\|I_i\|) = \log(I_i^c) - \log(\vec{R}_i^c)$ (for all channels $c$) and assuming a squared deviation around the log gradient magnitude, this translates into the following Gaussian MRF term on the reflectances

$$E_{ret}(\mathbf{r}) = \sum_{i \sim j} \left( \log(r_i) - \log(r_j) - g_{ij}(I)(\log(\|I_i\|) - \log(\|I_j\|)) \right)^2. \tag{5}$$

It remains to specify the classification function $g(I)$ for the image edges. In this work we adopt the Color Retinex version that has been proposed in [7]. For each pixel $i$ and a neighbor $j$ we compute the gradient of the intensity image and the gradient of the chromaticity change. If both gradients exceed a certain threshold ($\theta_g$ and $\theta_c$ resp.), the edge at $i, j$ is classified as being a "reflectance edge" and in this case $g_{ij}(I) = 1$. The two parameters which are the thresholds $\theta_g, \theta_c$ for the intensity and the chromaticity change are then estimated using leave-one-out-cross validation. It is worth noting that this term is qualitatively different from the smoothness prior on shading (4) even for pixels where $g_{ij}(I) = 0$. Here, the log-difference is penalized whereas the shading smoothness does also depend on the intensity values $\|I_i\|, \|I_j\|$. By setting $\mathrm{w}_{cl}, \mathrm{w}_s = 0$ in Eq. (2) we recover Color Retinex [7].

**Global Sparse Reflectance Prior ($E_{cl}$)** Motivated by the findings of [12] we include a term that acts as a global potential on the reflectances and favors the decomposition into some few reflectance clusters. We assume $C$ different reflectance clusters, each of which is denoted by $\tilde{R}_c, c \in \{1, \dots, C\}$. Every reflectance component $r_i$ belongs to one of the clusters and we denote its cluster membership with the variable $\alpha_i \in \{1, \dots, C\}$. This is summarized in the following energy term

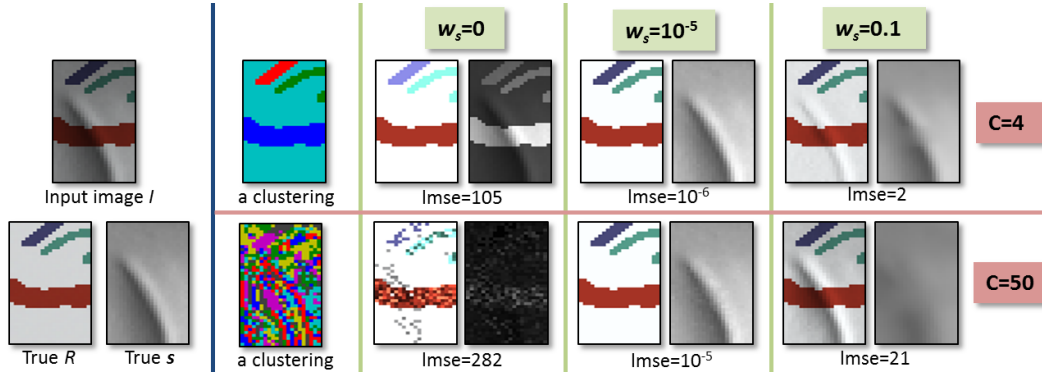

Figure 2: A crop from the image "panther". Left: input image $I$ and true decomposition $(R, \mathbf{s})$. Note, the colors in reflectance image (True $R$) have been modified on purpose such that there are exactly 4 different colors. The second column shows a clustering (here from the solution with $\mathrm{w}_s = 0$), where each cluster has an arbitrary color. The remaining columns show results with various settings for $C$ and $\mathrm{w}_s$ (left reflectance image, right shading image). Top row is the result for $C = 4$ and bottom row for $C = 50$ clusters, columns are results for $\mathrm{w}_s = 0, 10^{-5}$, and 0.1. Below the images is the corresponding LMSE score (described in Section 4.1). (Note, results are visually slightly different since the unknown overall global scaling factor $k$ is set differently, that is $I = (\mathbf{s}k)(1/kR)$.

$$E_{cl}(\mathbf{r}, \alpha) = \sum_{i=1}^{n} \| r_i \vec{R}_i - \tilde{R}_{\alpha_i} \|^2. \tag{6}$$

Here, both continuous $\mathbf{r}$ and discrete $\alpha$ variables are mixed. This represents a global potential, since the cluster means depend on the assignment of all pixels in the image. For fixed $\alpha$, this term is convex in $\mathbf{r}$ and for fixed $\mathbf{r}$ the optimum of $\alpha$ is a simple assignment problem. The cluster means $\tilde{R}_c$ are optimally determined given $\mathbf{r}$ and $\alpha$: $\tilde{R}_c = \frac{1}{|\{i:\alpha_i=c\}|} \sum_{i:\alpha_i=c} r_i \vec{R}_i$.

**Relationship between $E_{cl}$ and $E_s$**   The example in Figure 2 highlights the influence of the terms. We use a simplified model (2), namely $E_{cl} + \mathrm{w}_s E_s$, and vary $w_s$ as well as the number of clusters. Let us first consider the case where $\mathrm{w}_s = 0$ (third column). Independent of the clustering we get an imperfect result. This is expected since there is no constraint across clusters. Hence the shading within one cluster looks reasonable, but is not aligned across clusters. By adding a little bit of smoothing ($\mathrm{w}_s = 10^{-5}$; 4'th column), this problem is cured for both clusterings. It is very important to note that too many clusters (here C=50) do *not* affect the result very much. The reason is that enough clustering constraints are present to recover the variation in shading. If we were to give each pixel its own cluster this would no longer be true and we would get the trivial solution of $\mathbf{s} = 1$. Finally, results deteriorate when the smoothing term is too strong (last column $\mathrm{w}_s = 0.1$), since it prefers a constant shading. Note, that for this simple toy example the smoothness prior was not important, however for real images the best results are achieved by using a non-zero $\mathrm{w}_s$.

### 3.2   Optimization of (3)

The MAP problem (3) consists of both discrete and continuous variables and we solve it using coordinate descent. The entire algorithm is summarized in Algorithm 1. [4]

Given an initial value for $\alpha$ we have seen empirically that our function tends to yield same solutions, irrespective of the starting point $\mathbf{r}$. In order to be also robust with respect to this initial choice, we choose from a range of initial $\mathbf{r}$ values as described next. From these starting points we choose the one with the lowest objective value (energy) and its corresponding result.

---

**Algorithm 1** Coordinate Descent for solving (3)

1: Select $r^0$ as described in the text
2: $\alpha^0 \leftarrow$ K-Means clustering of $\{r_i^0 \vec{R}_i, i = 1, \dots, N\}$
3: $t \leftarrow 0$
4: **repeat**
5:   $r^{t+1} \leftarrow$ optimize (3) with $\alpha^t$ fixed
6:   $\tilde{R}_c = \sum_{i:\alpha_i=c} r_i \vec{R}_i / |\{i : \alpha_i = c\}|$
7:   $\alpha^{t+1} \leftarrow$ assign new cluster labels with $r^{t+1}$ fixed
8:   $t \leftarrow t + 1$
9: **until** $E(\mathbf{r}^{t-1}, \alpha^{t-1}) - E(\mathbf{r}^t, \alpha^t) < \theta$

| comment | $E_s$ | $E_{cl}$ | $E_{ret}$ | LOO-CV | best single | image opt. |
|---|---|---|---|---|---|---|
| Color Retinex | - | - | ✓ | 29.5 | 29.5 | 25.5 |
| no edge information | ✓ | ✓ | - | 30.0 | 30.6 | 18.2 |
| Col-Ret+ global term | - | ✓ | ✓ | 27.2 | 24.4 | 18.1 |
| full model | ✓ | ✓ | ✓ | 27.4 | 24.4 | 16.1 |

Table 1: Comparing the effect of including different terms. The column "best-single" is the parameter set that works best on all 16 images jointly, "image opt." is the result when choosing the parameters optimal for each image individually, based on ground truth information.

We have seen empirically that this procedure gives stable results. For instance, we virtually always achieve a lower energy compared to using the ground truth **r** as initial start point.

**Initialization of** $r$    It is reasonable to assume that the output has a fixed range, i.e. $0 \geq R_i^c, s_i \geq 1$ (for all $c, i$).[5] In particular, this is true for the data in [7]. From these constraints we can derive that $\|I_i\| \geq r_i \geq 3$. Given that, we use the following three starting points for **r**, by varying $\gamma \in \{0.3, 0.5, 0.7\}$: $r_i = \gamma \|I_i\| + 3(1 - \gamma)$. Additionally we choose the start point **r** $= 1$. From these four different initial settings we choose the result which corresponds to the lowest final energy.

**Initialization of** $\alpha$    Given an initial value for **r** we can compute the terms in Eq.(6) and use K-Means clustering to optimize it. We use the best solution from five restarts.

**Updating r**    for a given fixed $\alpha$ this is implemented using a conjugate gradient descent solver [1]. This typically converges in some few hundred iterations for the images used in the experiments.

**Updating $\alpha$**    for given **r** this is a simple assignment problem: $\alpha_i = \mathrm{argmin}_{c=1,...,C} \|r_i \vec{R}_i - \tilde{R}_c\|^2$.

## 4   Experiments

For the empirical evaluation we use the intrinsic image database that has been introduced in [7]. This dataset consists of 16 different images for all of which the ground truth shading and reflectance components are available. We refer to [7] for details on how this data was collected. Some of the images can be seen in Figure 3. In all experiments we compare against Color Retinex which was found to be the best performing method among those that take a single image as input. The method from [19] yields better results but requires multiple input images from different light variations.

### 4.1   Error metric

We report the performance of the algorithms using the two different error metrics that have been suggested by the creators of the database [7]. The first metric is the average of the localized mean squared error (LMSE) between the predicted and true shading and predicted and true reflectance image. [6] Since the LMSE vary considerably we also use the average rank of the algorithm.

### 4.2   Experimental set-up and parameter learning

All free parameters of the models, e.g. the weights $w_{cl}, w_s, w_r$ and the gradient thresholds $\theta_c, \theta_g$ have been chosen using a leave-one-out estimate (LOO-CV). Due to the high variance of the scores for the images we used the median error to score the parameters. Thus for image $i$ the parameter was chosen that leads to the lowest median error on all images except $i$. Additionally we record the best single parameter set that works well on all images, and the score that is obtained when using the optimal parameters on each image individually. Although the latter estimate involves knowing ground truth estimates we are interested in the lower bound of the performance, in an interactive scenario a user can provide additional information to achieve this, as in [4].

We select the parameters from the following ranges. Whenever used, we fix $w_{cl} = 1$ since it suffices to specify the relative difference between the parameters. For models using both the cluster and shading smoothness terms, we select from $w_s \in \{0.001, 0.01, 0.1\}$, for models that use the cluster and Color Retinex term $w_r \in \{0.001, 0.01, 0.1, 1, 10\}$. When all three terms are non-zero, we vary $w_s$ as above paired with $w_r \in \times\{0.1w_s, w_s, 10w_s\}$. The gradient thresholds are varied in $\theta_g, \theta_c \in \{0.075, 1\}$ which yields four possible configurations. The reflectance cluster count is varied in $C \in \{10, 50, 150\}$.

## 4.3 Comparison - Model variations

In a first set of experiments we investigate the influence of using combinations of the prior terms described in Section 3.1. The numerical results are summarized in Table 1.

The first observation is that the Color Retinex algorithm (1st row) performs about similar to the system using a shading smoothness prior together with the global factor $E_{cl}$ (2nd row). Note that the latter system does not use any gradient information for estimation. This confirms our intuition that the term $E_{cl}$ provides strong coupling information between reflectance components, as also discussed in Figure 2. The lower value for the image optimal setting of 18.2 compared to 25.5 for Color Retinex indicates that one would benefit from a better parameter estimate, i.e. the flexibility of this algorithm is higher. Equipping Color Retinex with the global reflectance term improves all recorded results (3rd vs 2nd row). Again it seems that the LOO-CV parameter estimation is more stable in this case. Combining all three parts (4th row) does not improve the results over Color Retinex with the reflectance prior. With knowledge about the optimal image parameter it yields a lower LMSE score (16.1 vs 18.1).

## 4.4 Comparison to Literature

In Table 2 we compare the numerical results of our method to other intrinsic image algorithms. We again include the single best parameter and image dependent optimal parameter set. Although those are positively biased and obviously decrease with model complexity we believe that they are informative, given the parameter estimation problems due to the diverse and small database. The full model using all terms $E_{cl}, E_s$ and $E_{cret}$ improves over all the compared methods that use only a single image as input, but SHE$^\times$ (see below). The difference in rank between (Col-Ret) and (full model) indicates that

|  | LOO-CV | rank | best single | im. opt. |
|---|---|---|---|---|
| TAP05 [17] | 56* | - | - | - |
| TAP06 [16] | 39* | - | - | - |
| SHE [14]$^+$ | n/a | n/a | 56.2 | n/a |
| SHE [15]$^\times$ | n/a | n/a | (20.4) | - |
| BAS [7] | 72.6 | 5.1 | 60.3 | 36.6 |
| Gray-Ret [7] | 40.7 | 4.9 | 40.7 | 28.9 |
| Col-Ret | 29.5 | 3.7 | 29.5 | 25.5 |
| full model | 27.4 | 3.0 | 24.4 | 16.1 |
| Weiss [19] | 21.5 | 2.7 | 21.5 | 21.5 |
| Weiss+Ret [7] | 16.4 | 1.7 | 16.4 | 15.0 |

Table 2: *Method comparison* with other intrinsic image algorithms also compared in [7]. Refer to Tab. 1 for a description of the quantities. Note that the last two methods from [19] use multiple input images. For entries '-' we had no individual results (and no code), the two numbers marked * are estimated from Fig4.a [7]. SHE$^+$ is our implementation. SHE$^\times$ Note that in [15] results were only given for 13 of 16 images from [7]. The additional data was kindly provided by authors.

the latter model is almost always better (direct comparison: 13 out of 16 images) than Color Retinex alone. The full model is even better on 6/16 images than the Weiss algorithm [19] that uses multiple images. Regarding the results of SHE$^\times$, we could not resolve with certainty whether the reported results should be compared as "best single" or "im.opt." (most parameters in [15] are common to all images, the strategy for setting $\lambda_{max}$ is not entirely specified). Assuming "best single" SHE$^\times$ is better in terms of LMSE, in direct comparison both models are better on 8/16 images. Comparing as an "im.opt." setting, our full model yields lower LMSE and is better on 12/16 images.

## 4.5 Visual Comparison

Additionally to the quantitative numbers we present some visual comparison in Figure 3, since the numbers not always reflect a visually pleasing results. For example note that the method BAS that either attributes all variations to shading ($\mathbf{r} = 1$) or to reflectance alone ($\mathbf{s} = 1$) already yields a LMSE of 36.6, if for every image the *optimal* choice between the two is made. Numerically this is better than [16, 17] and "Gray-Ret" with proper model selection. However the results of those algorithms are of course visually more pleasing. We have also tested our method on various other real-world images and results are visually similar to [15, 4]. Due to missing ground truth and lack of space we do not show them.

Figure 3 shows results with various models and settings. The "turtle" example (top three rows) shows the effect of the global term. Without the global term (Color Retinex with LOO-CV and image optimal) the result is imperfect. The key problem of Retinex is highlighted in the two zoom-in pictures with blue border (second column, left side). The upper one shows the detected edges in black. As expected the Retinex result has discontinuities at these edges, but over-smooths otherwise (lower picture). With a global term (remaining three results) the images look visually much better.

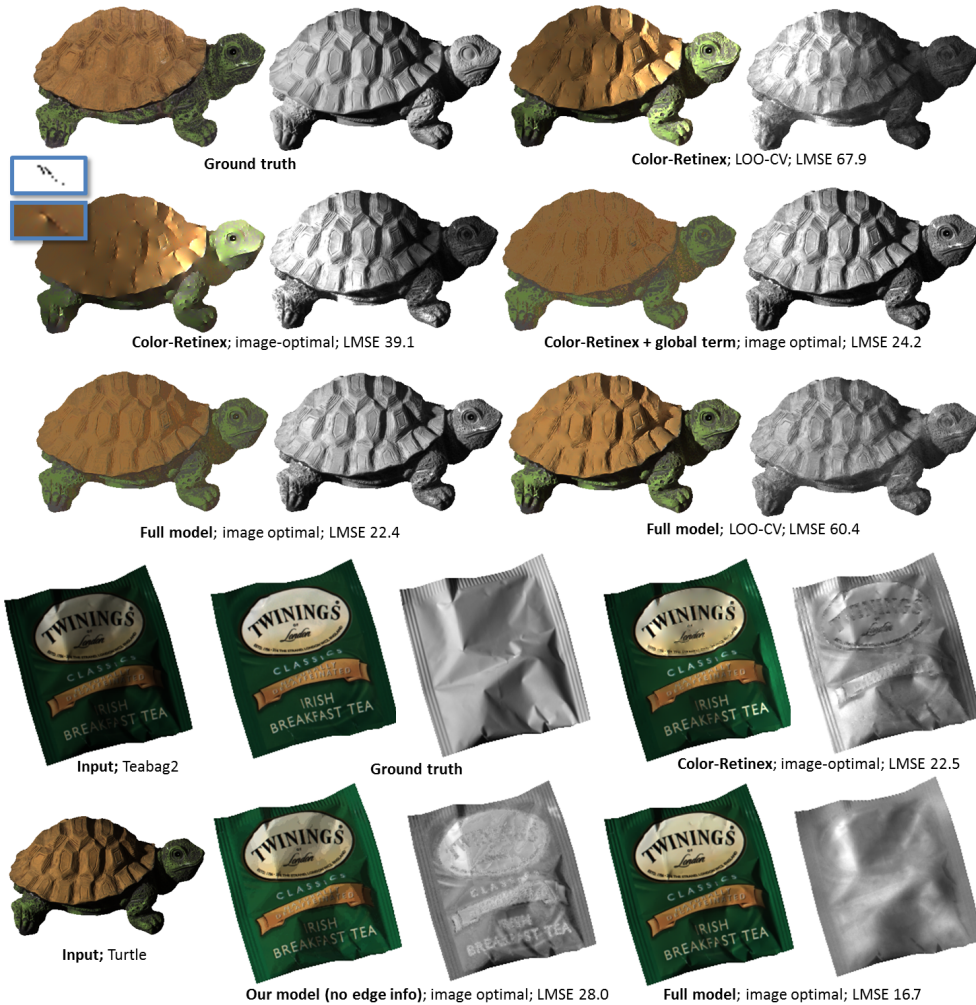

Figure 3: Various results obtained with different methods and settings (more in supplementary material); For each result: left reflectance image, right shading image

Note that the third row shows an extreme variation for the full model when switching from image optimal setting to LOO-CV setting. The example "teabag2" illustrates nicely the point that Color Retinex and our model without edge term (i.e. no Retinex term) achieve very complementary results. Our model without edges is sensitive to edge transitions, while Color Retinex has problems with fine details, e.g. the small text below "TWINGS". Combing all terms (full model) gives the best result with lowest LMSE score (16.4). Note, in this case we chose for both methods the image optimal settings to illustrate the potential of each model.

## 5   Discussion and Conclusion

We have introduced a new probabilistic model for intrinsic images that explicitly models the reflectance formation process. Several extensions are conceivable, e.g. one can relax the condition $I = \mathbf{s}R$ to allow deviations. Another refinement would be to replace the Gaussian cluster term with a color line term [12]. Building on the work of [5, 4] one can investigate various higher-order (patch-based) priors for both reflectance and shading.

A main concern is that in order to develop more advanced methods a larger and even more diverse database than the one of [7] is needed. This is especially true to enable learning of richer models such as Fields of Experts [13] or Gaussian CRFs [18]. We acknowledge the complexity of collecting ground truth data, but do believe that the creation of a new, much enlarged dataset, is a necessity for future progress in this field.

## Footnotes

[1]This problem can be overcome by utilizing a 3D vector for $s$, as done in [4], which we however do not consider in this work.

[2]Note, a gradient in chromaticity can only be caused by differently colored light sources, or inter-reflectance.

[3]We performed initial tests with this term. However, we found that it did not help to improve performance.

[4]Code available `http://people.tuebingen.mpg.de/mkiefel/projects/intrinsic`

[5]This assumption is violated if there is no global scalar $k$ such that $0 \geq (1/kR_i^c), (ks_i) \geq 1$.

[6]We multiply by 1000 for easier readability

# References

[1] www.gatsby.ucl.ac.uk/˜edward/code/minimize.

[2] H. G. Barrow and J. M. Tenenbaum. Recovering intrinsic scene characteristics from images. *Computer Vision Systems*, 1978.

[3] A. Blake. Boundary conditions for lightness computation in mondrian world. *Computer Vision, Graphics, and Image Processing*, 1985.

[4] A. Bousseau, S. Paris, and F. Durand. User assisted intrinsic images. *SIGGRAPH Asia*, 2009.

[5] W. T. Freeman, E. C. Pasztor, and O. T. Carmichael. Learning low-level vision. *International Journal of Computer Vision (IJCV)*, 2000.

[6] B. V. Funt, M. S. Drew, and M. Brockington. Recovering shading from color images. In *European Conference on Computer Vision (ECCV)*, 1992.

[7] R. Grosse, M. K. Johnson, E. H. Adelson, and W. T. Freeman. Ground-truth dataset and baseline evaluations for intrinsic image algorithms. In *International Conference on Computer Vision (ICCV)*, 2009.

[8] B. K. Horn. *Robot Vision*. MIT press, 1986.

[9] E. Land and J. McCann. Lightness and retinex theory. *Journal of the Optical Society of America*, 1971.

[10] A. Levin, D. Lischinski, and Y. Weiss. A closed form solution to natural image matting. *IEEE Transactions on Pattern Analysis and Machine Intelligence (PAMI)*, 30(2), 2008.

[11] Y.-H. W. Ming Shao. Recovering facial intrinsic images from a single input. *Lecture Notes in Computer Science*, 2009.

[12] I. Omer and M. Werman. Color lines: Image specific color representation. In *IEEE Conference on Computer Vision and Pattern Recognition (CVPR)*, 2004.

[13] S. Roth and M. J. Black. Fields of experts. *International Journal of Computer Vision (IJCV)*, 82(2):205–229, 2009.

[14] L. Shen, P. Tan, and S. Lin. Intrinsic image decomposition with non-local texture cues. In *IEEE Conference on Computer Vision and Pattern Recognition (CVPR)*, 2008.

[15] L. Shen and C. Yeo. Intrinsic images decomposition using a local and global sparse representation of reflectance. In *IEEE Conference on Computer Vision and Pattern Recognition (CVPR)*, 2011.

[16] M. Tappen, E. Adelson, and W. Freeman. Estimating intrinsic component images using non-linear regression. In *IEEE Conference on Computer Vision and Pattern Recognition (CVPR)*, 2006.

[17] M. Tappen, W. Freeman, and E. Adelson. Recovering intrinsic images from a single image. *IEEE Transactions on Pattern Analysis and Machine Intelligence (PAMI)*, 2005.

[18] M. Tappen, C. Liu, E. H. Adelson, and W. T.Freeman. Learning gaussian conditional random fields for low-level vision. In *IEEE Conference on Computer Vision and Pattern Recognition (CVPR)*, 2007.

[19] Y. Weiss. Deriving intrinsic images from image sequences. In *International Conference on Computer Vision (ICCV)*, 2001.

